# Learning Minimum Volume Sets

**Clayton Scott**
Statistics Department
Rice University
Houston, TX 77005
cscott@rice.edu

**Robert Nowak**
Electrical and Computer Engineering
University of Wisconsin
Madison, WI 53706
nowak@engr.wisc.edu

## Abstract

Given a probability measure $P$ and a reference measure $\mu$, one is often interested in the minimum $\mu$-measure set with $P$-measure at least $\alpha$. Minimum volume sets of this type summarize the regions of greatest probability mass of $P$, and are useful for detecting anomalies and constructing confidence regions. This paper addresses the problem of estimating minimum volume sets based on independent samples distributed according to $P$. Other than these samples, no other information is available regarding $P$, but the reference measure $\mu$ is assumed to be known. We introduce rules for estimating minimum volume sets that parallel the empirical risk minimization and structural risk minimization principles in classification. As in classification, we show that the performances of our estimators are controlled by the rate of uniform convergence of empirical to true probabilities over the class from which the estimator is drawn. Thus we obtain finite sample size performance bounds in terms of VC dimension and related quantities. We also demonstrate strong universal consistency and an oracle inequality. Estimators based on histograms and dyadic partitions illustrate the proposed rules.

## 1 Introduction

Given a probability measure $P$ and a reference measure $\mu$, the minimum volume set (MV-set) with mass at least $0 < \alpha < 1$ is

$$G_\alpha^* = \arg \min\{\mu(G) : P(G) \geq \alpha, G \text{ measurable}\}.$$

MV-sets summarize regions where the mass of $P$ is most concentrated. For example, if $P$ is a multivariate Gaussian distribution and $\mu$ is the Lebesgue measure, then the MV-sets are ellipsoids (see also Figure 1). Applications of minimum volume sets include outlier/anomaly detection, determining highest posterior density or multivariate confidence regions, tests for multimodality, and clustering. In comparison to the closely related problem of density level set estimation [1, 2], the minimum volume approach seems preferable in practice because the mass $\alpha$ is more easily specified than a level of a density. See [3, 4, 5] for further discussion of MV-sets.

This paper considers the problem of MV-set estimation using a training sample drawn from $P$, which in most practical settings is the only information one has

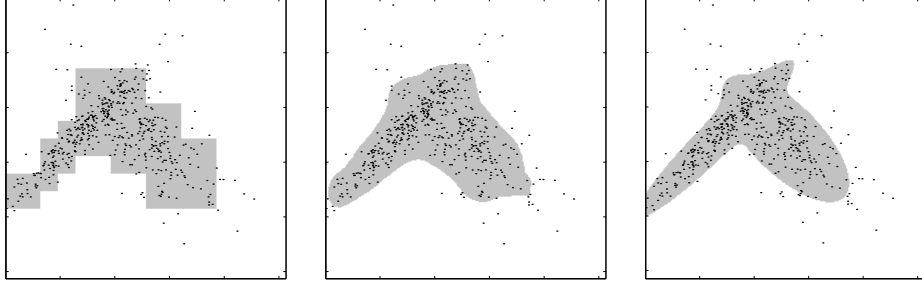

Figure 1: Gaussian mixture data, 500 samples, $\alpha = 0.9$. (Left and Middle) Minimum volume set estimates based on recursive dyadic partitions, discussed in Section 6. (Right) True MV set.

about $P$. The specifications to the estimation process are the significance level $\alpha$, the reference measure $\mu$, and a collection of candidate sets $\mathcal{G}$. All proofs, as well as additional results and discussion, may be found in [6] . To our knowledge, ours is the first work to establish finite sample bounds, an oracle inequality, and universal consistency for the MV-set estimation problem.

The methods proposed herein are primarily of theoretical interest, although they may be implemented effeciently for certain partition-based estimators as discussed later. As a more practical alternative, the MV-set problem may be reduced to Neyman-Pearson classification [7, 8] by simulating realizations from.

## 1.1 Notation

Let $(\mathcal{X}, \mathcal{B})$ be a measure space with $\mathcal{X} \subset \mathbb{R}^d$. Let $X$ be a random variable taking values in $\mathcal{X}$ with distribution $P$. Let $S = (X_1, \ldots, X_n)$ be an independent and identically distributed (IID) sample drawn according to $P$. Let $G$ denote a subset of $\mathcal{X}$, and let $\mathcal{G}$ be a collection of such subsets. Let $\widehat{P}$ denote the empirical measure based on $S$: $\widehat{P}(G) = (1/n) \sum_{i=1}^{n} I(X_i \in G)$. Here $I(\cdot)$ is the indicator function. Set

$$\mu_{\alpha}^{*} = \inf_{G} \{ \mu(G) : P(G) \geq \alpha \}, \tag{1}$$

where the inf is over all measurable sets. A minimum volume set, $G_{\alpha}^{*}$, is a minimizer of (1), when it exists. Let $\mathcal{G}$ be a class of sets. Given $\alpha \in (0, 1)$, denote $\mathcal{G}_{\alpha} = \{ G \in \mathcal{G} : P(G) \geq \alpha \}$, the collection of all sets in $\mathcal{G}$ with mass at least alpha. Define $\mu_{\mathcal{G}, \alpha} = \inf \{ \mu(G) : G \in \mathcal{G}_{\alpha} \}$ and $G_{\mathcal{G}, \alpha} = \arg \min \{ \mu(G) : G \in \mathcal{G}_{\alpha} \}$ when it exists. Thus $G_{\mathcal{G}, \alpha}$ is the best approximation to the MV-set $G_{\alpha}^{*}$ from $\mathcal{G}$. Existence and uniqueness of these and related quantities are discussed in [6] .

## 2 Minimum Volume Sets and Empirical Risk Minimization

In this section we introduce a procedure inspired by the empirical risk minimization (ERM) principle for classification. In classification, ERM selects a classifier from a fixed set of classifiers by minimizing the empirical error (risk) of a training sample. Vapnik and Chervonenkis established the basic theoretical properties of ERM (see [9, 10]), and we find similar properties in the minimum volume setting. In this and the next section we do not assume $P$ has a density with respect to $\mu$.

Let $\phi(G, S, \delta)$ be a function of $G \in \mathcal{G}$, the training sample $S$, and a confidence

parameter $\delta \in (0,1)$. Set $\widehat{\mathcal{G}}_\alpha = \{G \in \mathcal{G} : \widehat{P}(G) \geq \alpha - \phi(G, S, \delta)\}$ and

$$\widehat{G}_{\mathcal{G},\alpha} = \arg\min\{\mu(G) : G \in \widehat{\mathcal{G}}_\alpha\}. \tag{2}$$

We refer to the rule in (2) as MV-ERM because of the analogy with empirical risk minimization in classification. The quantity $\phi$ acts as a kind of "tolerance" by which the empirical mass estimate may deviate from the targeted value of $\alpha$. Throughout this paper we assume that $\phi$ satisfies the following.

**Definition 1.** *We say $\phi$ is a (distribution free) complexity penalty for $\mathcal{G}$ if and only if for all distributions $P$ and all $\delta \in (0,1)$,*

$$P^n\left(\left\{S : \sup_{G \in \mathcal{G}}\left(\left|P(G) - \widehat{P}(G)\right| - \phi(G, S, \delta)\right) > 0\right\}\right) \leq \delta.$$

Thus, $\phi$ controls the rate of uniform convergence of $\widehat{P}(G)$ to $P(G)$ for $G \in \mathcal{G}$. It is well known that the performance of ERM (for binary classification) relative to the performance of the best classifier in the given class is controlled by the uniform convergence of true to empirical probabilities. A similar result holds for MV-ERM.

**Theorem 1.** *If $\phi$ is a complexity penalty for $\mathcal{G}$, then*

$$P^n\left(\left(P(\widehat{G}_{\mathcal{G},\alpha}) < \alpha - 2\phi(\widehat{G}_{\mathcal{G},\alpha}, S, \delta)\right) \text{ or } \left(\mu(\widehat{G}_{\mathcal{G},\alpha}) > \mu_{\mathcal{G},\alpha}\right)\right) \leq \delta.$$

*Proof.* Consider the sets

$$
\begin{aligned}
\Theta_P &= \{S : P(\widehat{G}_{\mathcal{G},\alpha}) < \alpha - 2\phi(\widehat{G}_{\mathcal{G},\alpha}, S, \delta)\}, \\
\Theta_\mu &= \{S : \mu(\widehat{G}_{\mathcal{G},\alpha}) > \mu(G_{\mathcal{G},\alpha})\}, \\
\Omega_P &= \left\{S : \sup_{G \in \mathcal{G}}\left(\left|P(G) - \widehat{P}(G)\right| - \phi(G, S, \delta)\right) > 0\right\}.
\end{aligned}
$$

The result follows easily from the following lemma.

**Lemma 1.** *With $\Theta_P, \Theta_\mu$, and $\Omega_P$ defined as above and $\widehat{G}_{\mathcal{G},\alpha}$ as defined in (2) we have $\Theta_P \cup \Theta_\mu \subset \Omega_P$.*

The proof of this lemma (see [6]) follows closely the proof of Lemma 1 in [7]. This result may be understood by analogy with the result from classification that says $R(\widehat{f}) - \inf_{f \in \mathcal{F}} R(f) \leq 2 \sup_{f \in \mathcal{F}} |R(f) - \widehat{R}(f)|$ (see [10], Ch. 8). Here $R$ and $\widehat{R}$ are the true and empirical risks, $\widehat{f}$ is the empirical risk minimizer, and $\mathcal{F}$ is a set of classifiers. Just as this result relates uniform convergence bounds to empirical risk minimization in classification, so does Lemma 1 relate uniform convergence to the performance of MV-ERM. $\square$

The theorem above allows direct translation of uniform convergence results into performance guarantees for MV-ERM. Fortunately, many penalties (uniform convergence results) are known. We now give to important examples, although many others, such as the Rademacher penalty, are possible.

## 2.1 Example: VC Classes

Let $\mathcal{G}$ be a class of sets with VC dimension $V$, and define

$$\phi(G, S, \delta) = \sqrt{32\frac{V \log n + \log(8/\delta)}{n}}. \tag{3}$$

By a version of the VC inequality [10], we know that $\phi$ is a complexity penalty for $\mathcal{G}$, and therefore Theorem 1 applies. To view this result in perhaps a more recognizable way, let $\epsilon > 0$ and choose $\delta$ such that $2\phi(G, S, \delta) = \epsilon$. By inverting the relationship between $\delta$ and $\epsilon$, we have the following.

**Corollary 1.** *With the notation defined above,*

$$P^n\left(\left(P(\widehat{G}_{\mathcal{G},\alpha}) < \alpha - \epsilon\right) \text{ or } \left(\mu(\widehat{G}_{\mathcal{G},\alpha}) > \mu_{\mathcal{G},\alpha}\right)\right) \leq 8n^V e^{-n\epsilon^2/128}.$$

Thus, for any fixed $\epsilon > 0$, the probability of being within $\epsilon$ of the target mass $\alpha$ and being less than the target volume $\mu_{\mathcal{G},\alpha}$ approaches one exponentially fast as the sample size increases. This result may also be used to calculate a distribution free upper bound on the sample size needed to be within a given tolerance $\epsilon$ of $\alpha$ and with a given confidence $1 - \delta$. In particular, the sample size will grow no faster than a polynomial in $1/\epsilon$ and $1/\delta$, paralleling results for classification.

### 2.2 Example: Countable Classes

Suppose $\mathcal{G}$ is a countable class of sets. Assume that to every $G \in \mathcal{G}$ a number $[\![G]\!]$ is assigned such that $\sum_{G \in \mathcal{G}} 2^{-[\![G]\!]} \leq 1$. In light of the Kraft inequality for prefix codes, $[\![G]\!]$ may be defined as the codelength of a codeword for $G$ in a prefix code for $\mathcal{G}$. Let $\delta > 0$ and define

$$\phi(G, S, \delta) = \sqrt{\frac{[\![G]\!] \log 2 + \log(2/\delta)}{2n}}. \tag{4}$$

By Chernoff's bound together with the union bound, $\phi$ is a penalty for $\mathcal{G}$. Therefore Theorem 1 applies and we have obtained a result analogous to the Occam's Razor bound for classification.

As a special case, suppose $\mathcal{G}$ is finite and take $[\![G]\!] = \log_2 |\mathcal{G}|$. Setting $2\phi(G, S, \delta) = \epsilon$ and inverting the relationship between $\delta$ and $\epsilon$, we have

**Corollary 2.** *For the MV-ERM estimate $\widehat{G}_{\mathcal{G},\alpha}$ from a finite class $\mathcal{G}$*

$$P^n\left(\left(P(\widehat{G}_{\mathcal{G},\alpha}) < \alpha - \epsilon\right) \text{ or } \left(\mu(\widehat{G}_{\mathcal{G},\alpha}) > \mu_{\mathcal{G},\alpha}\right)\right) \leq 2|\mathcal{G}|e^{-n\epsilon^2/2}.$$

## 3 Consistency

A minimum volume set estimator is consistent if its volume and mass tend to the optimal values $\mu_\alpha^*$ and $\alpha$ as $n \to \infty$. Formally, define the error quantity

$$\mathcal{M}(G) := (\mu(G) - \mu_\alpha^*)_+ + (\alpha - P(G))_+,$$

where $(x)_+ = \max(x, 0)$. (Note that without the $(\cdot)_+$ operator, this would not be a meaningful error since one term could be negative and cause $\mathcal{M}$ to tend to zero, even if the other error term does not go to zero.) We are interested in MV-set estimators such that $\mathcal{M}(\widehat{G}_{\mathcal{G},\alpha})$ tends to zero as $n \to \infty$.

**Definition 2.** *A learning rule $\widehat{G}_{\mathcal{G},\alpha}$ is strongly consistent if $\lim_{n \to \infty} \mathcal{M}(\widehat{G}_{\mathcal{G},\alpha}) = 0$ with probability 1. If $\widehat{G}_{\mathcal{G},\alpha}$ is strongly consistent for every possible distribution of $X$, then $\widehat{G}_{\mathcal{G},\alpha}$ is strongly* universally *consistent.*

To see how consistency might result from MV-ERM, it helps to rewrite Theorem 1 as follows. Let $\mathcal{G}$ be fixed and let $\phi(G, S, \delta)$ be a penalty for $\mathcal{G}$. Then with probability at least $1 - \delta$, both

$$\mu(\widehat{G}_{\mathcal{G},\alpha}) - \mu_\alpha^* \leq \mu(G_{\mathcal{G},\alpha}) - \mu_\alpha^* \tag{5}$$

and

$$\alpha - P(\widehat{G}_{\mathcal{G},\alpha}) \leq 2\phi(\widehat{G}_{\mathcal{G},\alpha}, S, \delta) \tag{6}$$

hold. We refer to the left-hand side of (5) as the *excess volume* of the class $\mathcal{G}$ and the left-hand side of (6) as the *missing mass* of $\widehat{G}_{\mathcal{G},\alpha}$. The upper bounds on the right-hand sides are an approximation error and a stochastic error, respectively. The idea is to let $\mathcal{G}$ grow with $n$ so that both errors tend to zero as $n \to \infty$. If $\mathcal{G}$ does not change with $n$, universal consistency is impossible.

To have both stochastic and approximation errors tend to zero, we apply MV-ERM to a class $\mathcal{G}^k$ from a sequence of classes $\mathcal{G}^1, \mathcal{G}^2, \ldots$, where $k = k(n)$ grows with the sample size. Consider the estimator $\widehat{G}_{\mathcal{G}^k,\alpha}$.

**Theorem 2.** *Choose $k = k(n)$ and $\delta = \delta(n)$ such that $k(n) \to \infty$ as $n \to \infty$ and $\sum_{n=1}^{\infty} \delta(n) < \infty$. Assume the sequence of sets $\mathcal{G}^k$ and penalties $\phi_k$ satisfy*

$$\lim_{k \to \infty} \inf_{G \in \mathcal{G}_\alpha^k} \mu(G) = \mu_\alpha^* \tag{7}$$

*and*

$$\lim_{n \to \infty} \sup_{G \in \mathcal{G}_\alpha^k} \phi_k(G, S, \delta(n)) = o(1). \tag{8}$$

*Then $\widehat{G}_{\mathcal{G}^k,\alpha}$ is strongly universally consistent.*

The proof combines the Borel-Cantelli lemma and the distribution-free result of Theorem 1 with the stated assumptions. Examples satisfying the hypotheses of the theorem include families of VC classes with arbitrary approximating power (e.g., generalized linear discriminant rules with appropriately chosen basis functions and neural networks), and histogram rules. See [6] for further discussion.

## 4    Structural Risk Minimization and an Oracle Inequality

In the previous section the rate of convergence of the two errors to zero is determined by the choice of $k = k(n)$, which must be chosen a priori. Hence it is possible that the excess volume decays much more quickly than the missing mass, or vice versa. In this section we introduce a new rule called MV-SRM, inspired by the principle of structural risk minimization (SRM) from the theory of classification [11, 12], that automatically balances the two errors.

The result in this section is not distribution free. We assume

    **A1** $P$ has a density $f$ with respect to $\mu$.

    **A2** $G_\alpha^*$ exists and $P(G_\alpha^*) = \alpha$.

Under these assumptions (see [6] ) there exists $\gamma_\alpha > 0$ such that for any MV-set $G_\alpha^*$, $\{x : f(x) > \gamma_\alpha\} \subset G_\alpha^* \subset \{x : f(x) \geq \gamma_\alpha\}$.

Let $\mathcal{G}$ be a class of sets. Conceptualize $\mathcal{G}$ as a collection of sets of varying capacities, such as a union of VC classes or a union of finite classes. Let $\phi(G, S, \delta)$ be a penalty for $\mathcal{G}$. The MV-SRM principle selects the set

$$\widehat{G}_{\mathcal{G},\alpha} \;=\; \arg\min_{G \in \mathcal{G}} \left\{ \mu(G) + \phi(G, S, \delta) : \widehat{P}(G) \geq \alpha - \phi(G, S, \delta) \right\}. \tag{9}$$

Note that MV-SRM is different from MV-ERM because it minimizes a complexity penalized volume instead of simply the volume. We have the following.[1]

**Theorem 3.** *Let $\widehat{G}_{\mathcal{G},\alpha}$ be the MV-set estimator in (9). With probability at least $1 - \delta$ over the training sample S,*

$$\mathcal{M}(\widehat{G}_{\mathcal{G},\alpha}) \leq \left(1 + \frac{1}{\gamma_\alpha}\right) \inf_{G \in \mathcal{G}_\alpha} \left\{ \mu(G) - \mu_\alpha^* + 2\phi(G, S, \delta) \right\}. \qquad (10)$$

*Sketch of proof:* The proof is similar in some respects to oracle inequalities for classification. The key difference is in the form of the error term $\mathcal{M}(G) = (\mu(G) - \mu_\alpha^*)_+ + (\alpha - P(G))_+$. In classification both approximation and stochastic errors are positive, whereas with MV-sets the excess volume $\mu(G) - \mu_\alpha^*$ or missing mass $\alpha - P(G)$ could be negative. This necessitates the $(\cdot)_+$ operators, without which the error would not be meaningful as mentioned earlier. The proof considers three cases separately: (1) $\mu(\widehat{G}_{\mathcal{G},\alpha}) \geq \mu_\alpha^*$ and $P(\widehat{G}_{\mathcal{G},\alpha}) < \alpha$, (2) $\mu(\widehat{G}_{\mathcal{G},\alpha}) \geq \mu_\alpha^*$ and $P(\widehat{G}_{\mathcal{G},\alpha}) \geq \alpha$, and (3) $\mu(\widehat{G}_{\mathcal{G},\alpha}) < \mu_\alpha^*$ and $P(\widehat{G}_{\mathcal{G},\alpha}) < \alpha$. In the first case, both volume and mass errors are positive and the argument follows standard lines. The second case can be seen to follow easily from the first. The third case (which occurs most frequently in practice) is most involved and requires use of the fact that $\mu_\alpha^* - \mu_{\alpha-\epsilon}^* \leq \epsilon/\gamma_\alpha$ for $\epsilon > 0$, which can be deduced from basic properties of MV and density level sets. □

The oracle inequality says that MV-SRM performs about as well as the set chosen by an oracle to optimize the tradeoff between the stochastic and approximation errors. To illustrate the power of the oracle inequality, in [6] we demonstrate that MV-SRM applied to recursive dyadic partition-based estimators adapts optimally to the number of relevant features (unknown a priori).

## 5  Damping the Penalty

In Theorem 1, the reader may have noticed that MV-ERM does not equitably balance the volume error with the mass error. Indeed, with high probability, $\mu(\widehat{G}_{\mathcal{G},\alpha})$ is *less than* $\mu(G_{\mathcal{G},\alpha})$, while $P(\widehat{G}_{\mathcal{G},\alpha})$ is only guaranteed to be within $\phi(\widehat{G}_{\mathcal{G},\alpha})$ of $\alpha$. The net effect is that MV-ERM (and MV-SRM) underestimates the MV-set. Experimental comparisons have confirmed this to be the case [6] .

A minor modification of MV-ERM and MV-SRM leads to a more equitable distribution of error between the volume and mass, instead of having all the error reside in the mass term. The idea is simple: scale the penalty in the constraint by a damping factor $\nu < 1$. In the case of MV-SRM, the penalty in the objective function also needs to be scaled by $1 + \nu$. Moreover, the theoretical properties of these estimators stated above are retained (the statements, omitted here, are slightly more involved [6] ). Notice that in the case $\nu = 1$ we recover the original estimators. Also note that the above theorem encompasses the generalized quantile estimate of [3], which corresponds to $\nu = 0$. Thus we have finite sample size guarantees for that estimator to match Polonik's asymptotic analysis.

## 6  Experiments: Histograms and Trees

To gain some insight into the basic properties of our estimators, we devised some simple numerical experiments. In the case of histograms, MV-SRM can be implemented in a two step process. First, compute the MV-ERM estimate (a very simple procedure) for each $\mathcal{G}^k$, $k = 1, \ldots, K$, where $1/k$ is the bin-width. Second, choose the final estimate by minimizing the penalized volume of the MV-ERM estimates.

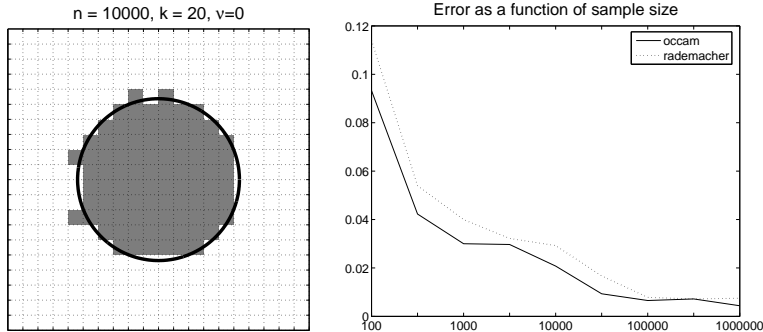

Figure 2: Results for histograms. (Left) A typical MV-ERM estimate with bin-width $1/20$, $\nu = 0$, and based on 10000 points. True MV-set indicated by solid line. (Right) The error of the MV-SRM estimate $\mathcal{M}(\widehat{G}_{\mathcal{G},\alpha})$ as a function of sample size when $\nu = 0$. The results indicated that the Occam's Razor bound is tighter and yields better performance than Rademacher.

We consider two penalties: one based on an Occam style bound, the other on the (conditional) Rademacher average. As a data set we consider $\mathcal{X} = [0,1]^2$, the unit square, and data generated by a two-dimensional truncated Gaussian distribution, centered at the point $(1/2, 1/2)$ and having spherical variance with parameter $\sigma = 0.15$. Other parameter settings are $\alpha = 0.8$, $K = 40$, and $\delta = 0.05$. All experiments were conducted at nine different sample sizes, logarithmically spaced from 100 to 1000000, and repeated 100 times. Results are summarized in Figure 2.

To illustrate the potential improvement offered by spatially adaptive partitioning methods, we consider a minimum volume set estimator based on recursive dyadic (quadsplit) partitions. We employ a penalty that is additive over the cells $A$ of the partition. The precise form of the penalty $\phi(A)$ for each cell is given in [6] , but loosely speaking it is proportional to the square-root of the ratio of the empirical mass of the cell to the sample size $n$. In this case, MV-SRM with $\nu = 0$ is

$$\min_{G \in \mathcal{G}^L} \sum_A [\mu(A)\ell(A) + \phi(A)] \quad \text{subject to} \quad \sum_A \widehat{P}(A)\ell(A) \geq \alpha \qquad (11)$$

where $\mathcal{G}^L$ is the collection of all partitions with dyadic cell sidelengths no smaller than $2^{-L}$ and $\ell(A) = 1$ if $A$ belongs to the candidate set and $\ell(A) = 0$ otherwise (see [6] for further details). Although directly optimization appears formidable, an efficient alternative is to consider the Lagrangian and conduct a bisection search over the Lagrange multiplier until the mass constraint is nearly achieved with equality (10 iterations is sufficient in practice). For each iteration, minimization of the Lagrangian can be performed very rapidly using standard tree pruning techniques.

An experimental demonstration of the dyadic partition estimator is depicted in Figure 1. In the experiments we employed a dyadic quadtree structure with $L = 8$ (i.e., cell sidelengths no smaller than $2^{-8}$) and pruned according to the theoretical penalty $\phi(A)$ formally defined in [6] weighted by a factor of $1/30$ (in practice the optimal weight could be found via cross-validation or other techniques). Figure 1 shows the results with data distributed according to a two-component Gaussian mixture distribution. This figure (middle image) additionally illustrates the improvement possible by "voting" over shifted partitions, which in principle is equivalent to constructing $2^L \times 2^L$ different trees, each based on a partition offset by an integer multiple of the base sidelength $2^{-L}$, and taking a majority vote over all the result-

ing set estimates to form the final estimate. This strategy mitigates the "blocky" structure due to the underlying dyadic partitions, and can be computed almost as rapidly as a single tree estimate (within a factor of $L$) due to the large amount of redundancy among trees. The actual running time was one to two seconds.

# 7 Conclusions

In this paper we propose two rules, MV-ERM and MV-SRM, for estimation of minimum volume sets. Our theoretical analysis is made possible by relating the performance of these rules to the uniform convergence properties of the class of sets from which the estimate is taken. Ours are the first known results to feature finite sample bounds, an oracle inequality, and universal consistency.

### Acknowledgements

The authors thank Ercan Yildiz and Rebecca Willett for their assistance with the experiments involving dyadic trees.

## Footnotes

[1]Although the value of $1/\gamma_\alpha$ is in practice unknown, it can be bounded by $1/\gamma_\alpha \leq (1 - \mu_\alpha^*)/(1 - \alpha) \leq 1/(1 - \alpha)$. This follows from the bound $1 - \alpha \leq \gamma_\alpha \cdot (1 - \mu_\alpha^*)$ on the mass outside the minimum volume set.

### References

[1] I. Steinwart, D. Hush, and C. Scovel, "A classification framework for anomaly detection," *J. Machine Learning Research*, vol. 6, pp. 211–232, 2005.

[2] S. Ben-David and M. Lindenbaum, "Learning distributions by their density levels – a paradigm for learning without a teacher," *Journal of Computer and Systems Sciences*, vol. 55, no. 1, pp. 171–182, 1997.

[3] W. Polonik, "Minimum volume sets and generalized quantile processes," *Stochastic Processes and their Applications*, vol. 69, pp. 1–24, 1997.

[4] G. Walther, "Granulometric smoothing," *Ann. Stat.*, vol. 25, pp. 2273–2299, 1997.

[5] B. Schölkopf, J. Platt, J. Shawe-Taylor, A. Smola, and R. Williamson, "Estimating the support of a high-dimensional distribution," *Neural Computation*, vol. 13, no. 7, pp. 1443–1472, 2001.

[6] C. Scott and R. Nowak, "Learning minimum volume sets," UW-Madison, Tech. Rep. ECE-05-2, 2005. [Online]. Available: http://www.stat.rice.edu/~cscott

[7] A. Cannon, J. Howse, D. Hush, and C. Scovel, "Learning with the Neyman-Pearson and min-max criteria," Los Alamos National Laboratory, Tech. Rep. LA-UR 02-2951, 2002. [Online]. Available: http://www.c3.lanl.gov/~kelly/ml/pubs/2002_minmax/paper.pdf

[8] C. Scott and R. Nowak, "A Neyman-Pearson approach to statistical learning," *IEEE Trans. Inform. Theory*, 2005, (in press).

[9] V. Vapnik, *Statistical Learning Theory.* New York: Wiley, 1998.

[10] L. Devroye, L. Györfi, and G. Lugosi, *A Probabilistic Theory of Pattern Recognition.* New York: Springer, 1996.

[11] V. Vapnik, *Estimation of Dependencies Based on Empirical Data.* New York: Springer-Verlag, 1982.

[12] G. Lugosi and K. Zeger, "Concept learning using complexity regularization," *IEEE Trans. Inform. Theory*, vol. 42, no. 1, pp. 48–54, 1996.